# Ordinal Regression by Extended Binary Classification

**Ling Li**
Learning Systems Group
California Institute of Technology
ling@caltech.edu

**Hsuan-Tien Lin**
Learning Systems Group
California Institute of Technology
htlin@caltech.edu

## Abstract

We present a reduction framework from ordinal regression to binary classification based on extended examples. The framework consists of three steps: extracting extended examples from the original examples, learning a binary classifier on the extended examples with any binary classification algorithm, and constructing a ranking rule from the binary classifier. A weighted 0/1 loss of the binary classifier would then bound the mislabeling cost of the ranking rule. Our framework allows not only to design good ordinal regression algorithms based on well-tuned binary classification approaches, but also to derive new generalization bounds for ordinal regression from known bounds for binary classification. In addition, our framework unifies many existing ordinal regression algorithms, such as perceptron ranking and support vector ordinal regression. When compared empirically on benchmark data sets, some of our newly designed algorithms enjoy advantages in terms of both training speed and generalization performance over existing algorithms, which demonstrates the usefulness of our framework.

## 1 Introduction

We work on a type of supervised learning problems called *ranking* or *ordinal regression*, where examples are labeled by an ordinal scale called the *rank*. For instance, the rating that a customer gives on a movie might be one of do-not-bother, only-if-you-must, good, very-good, and run-to-see. The ratings have a natural order, which distinguishes ordinal regression from general multiclass classification.

Recently, many algorithms for ordinal regression have been proposed from a machine learning perspective. For instance, Crammer and Singer [1] generalized the online perceptron algorithm with multiple thresholds to do ordinal regression. In their approach, a perceptron maps an input vector to a latent potential value, which is then thresholded to obtain a rank. Shashua and Levin [2] proposed new support vector machine (SVM) formulations to handle multiple thresholds. Some other formulations were studied by Rajaram et al. [3] and Chu and Keerthi [4]. All these algorithms share a common property: they are modified from well-known binary classification approaches.

Since binary classification is much better studied than ordinal regression, a general framework to systematically reduce the latter to the former can introduce two immediate benefits. First, well-tuned binary classification approaches can be readily transformed into good ordinal regression algorithms, which saves immense efforts in design and implementation. Second, new generalization bounds for ordinal regression can be easily derived from known bounds for binary classification, which saves tremendous efforts in theoretical analysis.

In this paper, we propose such a reduction framework. The framework is based on extended examples, which are extracted from the original examples and a given mislabeling cost matrix. The binary classifier trained from the extended examples can then be used to construct a ranking rule. We prove that the mislabeling cost of the ranking rule is bounded by a weighted 0/1 loss of the

binary classifier. Hence, binary classifiers that generalize well could introduce ranking rules that generalize well. The advantages of the framework in algorithmic design and in theoretical analysis are both demonstrated in the paper. In addition, we show that our framework provides a unified view for many existing ordinal regression algorithms. The experiments on some benchmark data sets validate the usefulness of our framework in practice.

The paper is organized as follows. In Section 2, we introduce our reduction framework. An unified view of some existing algorithms based on the framework is discussed in Section 3. Theoretical guarantee on the reduction, including derivations of new generalization bounds for ordinal regression, is provided in Section 4. We present experimental results of several new algorithms in Section 5, and conclude in Section 6.

## 2   The reduction framework

In an ordinal regression problem, an example $(\mathbf{x}, y)$ is composed of an input vector $\mathbf{x} \in \mathcal{X}$ and an ordinal label (i.e., rank) $y \in \mathcal{Y} = \{1, 2, \ldots, K\}$. Each example is assumed to be drawn i.i.d. from some unknown distribution $P(\mathbf{x}, y)$ on $\mathcal{X} \times \mathcal{Y}$. The generalization error of a ranking rule $r \colon \mathcal{X} \to \mathcal{Y}$ is then defined as

$$C(r, P) \stackrel{\text{def}}{=} \mathbb{E}_{(\mathbf{x}, y) \sim P} \mathcal{C}_{y, r(\mathbf{x})} \,,$$

where $\mathcal{C}$ is a $K \times K$ cost matrix with $\mathcal{C}_{y,k}$ being the cost of predicting an example $(\mathbf{x}, y)$ as rank $k$. Naturally we assume $\mathcal{C}_{y,y} = 0$ and $\mathcal{C}_{y,k} > 0$ for $k \neq y$. Given a training set $\mathcal{S} = \{(\mathbf{x}_n, y_n)\}_{n=1}^{N}$ containing $N$ examples, the goal is to find a ranking rule $r$ that generalizes well, i.e., associates with a small $C(r, P)$.

The setting above looks similar to that of a multiclass classification problem, except that the ranks are ordered. The ordinal information can be interpreted in several ways. In statistics, the information is assumed to reflect a stochastic ordering on the conditional distributions $P(y \leq k \mid \mathbf{x})$ [5]. Another interpretation is that the mislabeling cost depends on the "closeness" of the prediction. Consider an example $(\mathbf{x}, 4)$ with $r_1(\mathbf{x}) = 3$ and $r_2(\mathbf{x}) = 1$. The rule $r_2$ should pay more for the erroneous prediction than the rule $r_1$. Thus, we generally want each row of $\mathcal{C}$ to be *V-shaped*. That is, $\mathcal{C}_{y,k-1} \geq \mathcal{C}_{y,k}$ if $k \leq y$ and $\mathcal{C}_{y,k} \leq \mathcal{C}_{y,k+1}$ if $k \geq y$.

A simple $\mathcal{C}$ with V-shaped rows is the *classification cost matrix*, with entries $\mathcal{C}_{y,k} = [\![y \neq k]\!]$.[1] The classification cost is widely used in multiclass classification. However, because the cost is invariant for all kinds of mislabelings, the ordinal information is not taken into account. The *absolute cost matrix*, which is defined by $\mathcal{C}_{y,k} = |y - k|$, is a popular choice that better reflects the ordering preference. Its rows are not only V-shaped, but also *convex*. That is, $\mathcal{C}_{y,k+1} - \mathcal{C}_{y,k} \geq \mathcal{C}_{y,k} - \mathcal{C}_{y,k-1}$ for $1 < k < K$. The convex rows encode a stronger preference in making the prediction "close."

In this paper, we shall always assume that the ordinal regression problem under study comes with a cost matrix of V-shaped rows, and discuss how to reduce the ordinal regression problem to a binary classification problem. Some of the results may require the rows to be convex.

### 2.1   Reducing ordinal regression to binary classification

The ordinal information allows ranks to be compared. Consider, for instance, that we want to know how good a movie $\mathbf{x}$ is. An associated question would be: "is the rank of $\mathbf{x}$ greater than $k$?" For a fixed $k$, such a question is exactly a binary classification problem, and the rank of $\mathbf{x}$ can be determined by asking multiple questions for $k = 1, 2$, until $(K - 1)$. Frank and Hall [6] proposed to solve each binary classification problem independently and combine the binary outputs to a rank. Although their approach is simple, the generalization performance using the combination step cannot be easily analyzed.

Our framework works differently. First, all the binary classification problems are solved jointly to obtain a single binary classifier. Second, a simpler step is used to convert the binary outputs to a rank, and generalization analysis can immediately follow.

Assume that $f_{\mathrm{b}}(\mathbf{x}, k)$ is a binary classifier for all the associated questions above. *Consistent* answers would be $f_{\mathrm{b}}(\mathbf{x}, k) = 1$ ("yes") for $k = 1$ until $(y' - 1)$ for some $y'$, and 0 ("no") afterwards. Then, a reasonable ranking rule based on the binary answers is $r(\mathbf{x}) = y' = 1 + \min\{k\colon f_{\mathrm{b}}(\mathbf{x}, k) = 1\}$. Equivalently,

$$r(\mathbf{x}) \stackrel{\text{def}}{=} 1 + \sum_{k=1}^{K-1} f_{\mathrm{b}}(\mathbf{x}, k).$$

Although the definition can be flexibly applied even when $f_{\mathrm{b}}$ is not consistent, a consistent $f_{\mathrm{b}}$ is usually desired in order to introduce a good ranking rule $r$.

Furthermore, the ordinal information can help to model the relative confidence in the binary outputs. That is, when $k$ is farther from the rank of $\mathbf{x}$, the answer $f_{\mathrm{b}}(\mathbf{x}, k)$ should be more confident. The confidence can be modeled by a real-valued function $f\colon \mathcal{X} \times \{1, 2, \ldots, K - 1\} \to \mathbb{R}$, with $f_{\mathrm{b}}(\mathbf{x}, k) = [\![ f(\mathbf{x}, k) > 0 ]\!]$ and the confidence encoded in the magnitude of $f$. Accordingly,

$$r(\mathbf{x}) \stackrel{\text{def}}{=} 1 + \sum_{k=1}^{K-1} [\![ f(\mathbf{x}, k) > 0 ]\!]. \tag{1}$$

The ordinal information would naturally require $f$ to be *rank-monotonic*, i.e., $f(\mathbf{x}, 1) \geq f(\mathbf{x}, 2) \geq \cdots \geq f(\mathbf{x}, K - 1)$ for every $\mathbf{x}$. Note that a rank-monotonic function $f$ introduces consistent answers $f_{\mathrm{b}}$. Again, although the construction (1) can be applied to cases where $f$ is not rank-monotonic, a rank-monotonic $f$ is usually desired.

When $f$ is rank-monotonic, we have $f(\mathbf{x}, k) > 0$ for $k < r(\mathbf{x})$, and $f(\mathbf{x}, k) \leq 0$ for $k \geq r(\mathbf{x})$. Thus the cost of the ranking rule $r$ on an example $(\mathbf{x}, y)$ is

$$\mathcal{C}_{y, r(\mathbf{x})} = \sum_{k=r(\mathbf{x})}^{K-1} (\mathcal{C}_{y,k} - \mathcal{C}_{y,k+1}) + \mathcal{C}_{y,K} = \sum_{k=1}^{K-1} (\mathcal{C}_{y,k} - \mathcal{C}_{y,k+1}) [\![ f(\mathbf{x}, k) \leq 0 ]\!] + \mathcal{C}_{y,K}. \tag{2}$$

Define the extended examples $(\mathbf{x}^{(k)}, y^{(k)})$ with weights $w_{y,k}$ as

$$\mathbf{x}^{(k)} = (\mathbf{x}, k), \quad y^{(k)} = 2[\![ k < y ]\!] - 1, \quad w_{y,k} = |\mathcal{C}_{y,k} - \mathcal{C}_{y,k+1}|. \tag{3}$$

Because row $y$ in $\mathcal{C}$ is V-shaped, the binary variable $y^{(k)}$ equals the sign of $(\mathcal{C}_{y,k} - \mathcal{C}_{y,k+1})$ if the latter is not zero. Continuing from (2),

$$\mathcal{C}_{y, r(\mathbf{x})} = \sum_{k=1}^{y-1} w_{y,k} \cdot y^{(k)} [\![ f(\mathbf{x}^{(k)}) \leq 0 ]\!] + \sum_{k=y}^{K-1} w_{y,k} \cdot y^{(k)} \left( 1 - [\![ f(\mathbf{x}^{(k)}) > 0 ]\!] \right) + \mathcal{C}_{y,K}$$

$$= \sum_{k=1}^{y-1} w_{y,k} [\![ y^{(k)} f(\mathbf{x}^{(k)}) \leq 0 ]\!] + \mathcal{C}_{y,y} + \sum_{k=y}^{K-1} w_{y,k} [\![ y^{(k)} f(\mathbf{x}^{(k)}) < 0 ]\!]$$

$$\leq \sum_{k=1}^{K-1} w_{y,k} [\![ y^{(k)} f(\mathbf{x}^{(k)}) \leq 0 ]\!]. \tag{4}$$

Inequality (4) shows that the cost of $r$ on example $(\mathbf{x}, y)$ is bounded by a weighted 0/1 loss of $f$ on the extended examples. It becomes an equality if the degenerate case $f(\mathbf{x}^{(k)}) = 0$ does not happen.

When $f$ is not rank-monotonic but row $y$ of $\mathcal{C}$ is convex, the inequality (4) could be alternatively proved from

$$\sum_{k=r(\mathbf{x})}^{K-1} (\mathcal{C}_{y,k} - \mathcal{C}_{y,k+1}) \leq \sum_{k=1}^{K-1} (\mathcal{C}_{y,k} - \mathcal{C}_{y,k+1}) [\![ f(\mathbf{x}^{(k)}) \leq 0 ]\!].$$

The inequality above holds because $(\mathcal{C}_{y,k} - \mathcal{C}_{y,k+1})$ is decreasing due to the convexity, and there are exactly $(r(\mathbf{x}) - 1)$ zeros and $(K - r(\mathbf{x}))$ ones in the values of $[\![ f(\mathbf{x}^{(k)}) \leq 0 ]\!]$ in (1).

Altogether, our reduction framework consists of the following steps: we first use (3) to transform all training examples $(\mathbf{x}_n, y_n)$ to extended examples $(\mathbf{x}_n^{(k)}, y_n^{(k)})$ with weights $w_{y_n,k}$ (also denoted as $w_n^{(k)}$). All the extended examples would then be jointly learned by a binary classifier $f$ with confidence outputs, aiming at a low weighted 0/1 loss. Finally, a ranking rule $r$ is constructed from $f$ using (1). The cost bound in (4) leads to the following theorem.

**Theorem 1 (reduction)** *An ordinal regression problem with a V-shaped cost matrix $\mathcal{C}$ can be reduced to a binary classification problem with the extended examples in (3) and the ranking rule $r$ in (1). If $f$ is rank-monotonic or every row of $\mathcal{C}$ is convex, for any example $(\mathbf{x}, y)$ and its extended examples $(\mathbf{x}^{(k)}, y^{(k)})$, the weighted sum of the 0/1 loss of $f(\mathbf{x}^{(k)})$ bounds the cost of $r(\mathbf{x})$.*

## 2.2 Thresholded model

From Theorem 1 and the illustrations above, a rank-monotonic $f$ is preferred for our framework. A popular approach to obtain such a function $f$ is to use a thresholded model [1, 4, 5, 7]:

$$f(\mathbf{x}, k) = g(\mathbf{x}) - \theta_k.$$

As long as the threshold vector $\boldsymbol{\theta}$ is *ordered*, i.e., $\theta_1 \leq \theta_2 \leq \cdots \leq \theta_{K-1}$, the function $f$ is rank-monotonic. The question is then, "when can a binary classification algorithm return ordered thresholds?" A mild but sufficient condition is shown as follows.

**Theorem 2 (ordered thresholds)** *If every row of the cost matrix is convex, and the binary classification algorithm minimizes the loss*

$$\Lambda(g) + \sum_{n=1}^{N} \sum_{k=1}^{K-1} w_n^{(k)} \cdot \ell\left(y_n^{(k)}\left(g(\mathbf{x}_n) - \theta_k\right)\right), \tag{5}$$

*where $\ell(\rho)$ is non-increasing in $\rho$, there exists an optimal solution $(g^*, \boldsymbol{\theta}^*)$ such that $\boldsymbol{\theta}^*$ is ordered.*

PROOF For an optimal solution $(g, \boldsymbol{\theta})$, assume that $\theta_k > \theta_{k+1}$ for some $k$. We shall prove that switching $\theta_k$ and $\theta_{k+1}$ would not increase the objective value of (5). First, consider an example with $y_n = k + 1$. Since $y_n^{(k)} = 1$ and $y_n^{(k+1)} = -1$, switching the thresholds changes the objective value by

$$w_n^{(k)}\left[\ell(g(\mathbf{x}_n) - \theta_{k+1}) - \ell(g(\mathbf{x}_n) - \theta_k)\right] + w_n^{(k+1)}\left[\ell(\theta_k - g(\mathbf{x}_n)) - \ell(\theta_{k+1} - g(\mathbf{x}_n))\right]. \tag{6}$$

Because $\ell(\rho)$ is non-increasing, the change is non-positive.

For an example with $y_n < k + 1$, we have $y_n^{(k)} = y_n^{(k+1)} = -1$. The change in the objective is

$$(w_n^{(k)} - w_n^{(k+1)})\left[\ell(\theta_{k+1} - g(\mathbf{x}_n)) - \ell(\theta_k - g(\mathbf{x}_n))\right].$$

Note that row $y_n$ of the cost matrix being convex leads to $w_n^{(k)} \leq w_n^{(k+1)}$ if $y_n < k + 1$. Since $\ell(\rho)$ is non-increasing, the change above is also non-positive. The case for examples with $y_n > k + 1$ is similar and the change there is also non-positive.

Thus, by switching adjacent pairs of strictly decreasing thresholds, we can actually obtain a solution $(g^*, \boldsymbol{\theta}^*)$ with a smaller or equal objective value in (5), and $g^* = g$. The optimality of $(g, \boldsymbol{\theta})$ shows that $(g^*, \boldsymbol{\theta}^*)$ is also optimal. ∎

Note that if $\ell(\rho)$ is strictly decreasing for $\rho < 0$, and there are training examples for every rank, the change (6) is strictly negative. Thus, the optimal $\boldsymbol{\theta}^*$ for any $g^*$ is always ordered.

## 3 Algorithms based on the framework

So far the reduction works only by assuming that $\mathbf{x}^{(k)} = (\mathbf{x}, k)$ is a pair understandable by $f$. Actually, any lossless encoding from $(\mathbf{x}, k)$ to a vector can be used to encode the pair. With proper choices of the cost matrix, the encoding scheme of $(\mathbf{x}, k)$, and the binary learning algorithm, many existing ordinal regression algorithms can be unified in our framework. In this section, we will briefly discuss some of them. It happens that a simple encoding scheme for $(\mathbf{x}, k)$ via a coding matrix $\mathbf{E}$ of $(K - 1)$ rows works for all these algorithms. To form $\mathbf{x}^{(k)}$, the vector $\mathbf{e}_k$, which denotes the $k$-th row of $\mathbf{E}$, is appended after $\mathbf{x}$. We will mostly work with $\mathbf{E} = \gamma \mathbf{I}_{K-1}$, where $\gamma$ is a positive scalar and $\mathbf{I}_{K-1}$ is the $(K - 1) \times (K - 1)$ identity matrix.

### 3.1 Perceptron-based algorithms

The perceptron ranking (PRank) algorithm proposed by Crammer and Singer [1] is an online ordinal regression algorithm that employs the thresholded model with $f(\mathbf{x}, k) = \langle \mathbf{u}, \mathbf{x} \rangle - \theta_k$. Whenever a training example is not predicted correctly, the current $\mathbf{u}$ and $\boldsymbol{\theta}$ are updated in a way similar to the perceptron learning rule [8]. The algorithm was proved to keep an ordered $\boldsymbol{\theta}$, and a mistake bound was also proposed [1].

With the simple encoding scheme $\mathbf{E} = \mathbf{I}_{K-1}$, we can see that $f(\mathbf{x}, k) = \langle (\mathbf{u}, -\boldsymbol{\theta}), \mathbf{x}^{(k)} \rangle$. Thus, when the absolute cost matrix is taken and a modified perceptron learning rule[2] is used as the underlying binary classification algorithm, the PRank algorithm is a specific instance of our framework. The orderliness of the thresholds is guaranteed by Theorem 2, and the mistake bound is a direct application of the well-known perceptron mistake bound (see for example Freund and Schapire [8]). Our framework not only simplifies the derivation of the mistake bound, but also allows the use of other perceptron algorithms, such as a batch-mode algorithm rather than an online one.

### 3.2 SVM-based algorithms

SVM [9] can be thought as a generalized perceptron with a kernel that computes the inner product on transformed input vectors $\phi(\mathbf{x})$. For the extended examples $(\mathbf{x}, k)$, we can suitably define the extended kernel as the original kernel plus the inner product between the extensions,

$$\mathcal{K}\left((\mathbf{x}, k), (\mathbf{x}', k')\right) = \langle \phi(\mathbf{x}), \phi(\mathbf{x}') \rangle + \langle \mathbf{e}_k, \mathbf{e}_{k'} \rangle.$$

Then, several SVM-based approaches for ordinal regression are special instances of our framework. For example, the approach of Rajaram et al. [3] is equivalent to using the classification cost matrix, the coding matrix $\mathbf{E}$ defined with $e_{k,i} = \gamma \cdot [\![ k \leq i ]\!]$ for some $\gamma > 0$, and the hard-margin SVM.

When $\mathbf{E} = \gamma \mathbf{I}_{K-1}$ and the traditional soft-margin SVM are used in our framework, the binary classifier $f(\mathbf{x}, k)$ has the form $\langle \mathbf{u}, \phi(\mathbf{x}) \rangle - \theta_k - b$, and can be obtained by solving

$$\min_{\mathbf{u}, \boldsymbol{\theta}, b} \|\mathbf{u}\|^2 + \|\boldsymbol{\theta}\|^2 / \gamma^2 + \kappa \sum_{n=1}^{N} \sum_{k=1}^{K-1} w_n^{(k)} \max \left\{ 0, 1 - y_n^{(k)} \left( \langle \mathbf{u}, \phi(\mathbf{x}_n) \rangle - \theta_k - b \right) \right\}. \quad (7)$$

The explicit (SVOR-EXP) and implicit (SVOR-IMC) approaches of Chu and Keerthi [4] can be regarded as instances of our framework with a modified soft-margin SVM formulation (since they excluded the term $\|\boldsymbol{\theta}\|^2 / \gamma^2$ and added some constraints on $\boldsymbol{\theta}$). Thus, many of their results can be alternatively explained with our reduction framework. For example, their proof for ordered $\boldsymbol{\theta}$ of SVOR-IMC is implied from Theorem 2. In addition, they found that SVOR-EXP performed better in terms of the classification cost, and SVOR-IMC preceded in terms of the absolute cost. This finding can also be explained by reduction: SVOR-EXP is an instance of our framework using the classification cost and SVOR-IMC comes from using the absolute cost.

Note that Chu and Keerthi paid much effort in designing and implementing suitable optimizers for their modified formulation. If the unmodified soft-margin SVM (7) is directly used in our framework with the absolute cost, we obtain a new support vector ordinal regression formulation.[3] From Theorem 2, the thresholds $\boldsymbol{\theta}$ would be ordered. The dual of (7) can be easily solved with state-of-the-art SVM optimizers, and the formulations of Chu and Keerthi can be approximated by setting $\gamma$ to a large value. As we shall see in Section 5, even a simple setting of $\gamma = 1$ performs similarly to the approaches of Chu and Keerthi in practice.

## 4 Generalization bounds

With the extended examples, new generalization bounds can be derived for ordinal regression problems with any cost matrix. A simple result that comes immediately from (4) is:

**Theorem 3 (reduction of generalization error)** *Let $c_y = \mathcal{C}_{y,1} + \mathcal{C}_{y,K}$ and $c = \max_y c_y$. If $f$ is rank-monotonic or every row of $\mathcal{C}$ is convex, there exists a distribution $\hat{P}$ on $(\mathbf{X}, Y)$, where $\mathbf{X}$ contains the encoding of $(\mathbf{x}, k)$ and $Y$ is a binary label, such that*

$$\mathop{\mathbb{E}}_{(\mathbf{x},y)\sim P} \mathcal{C}_{y,r(\mathbf{x})} \leq c \cdot \mathop{\mathbb{E}}_{(\mathbf{X},Y)\sim \hat{P}} [\![ Yf(\mathbf{X}) \leq 0 ]\!].$$

PROOF We prove by constructing $\hat{P}$. Given the conditions, following (4), we have

$$\mathcal{C}_{y,r(\mathbf{x})} \leq \sum_{k=1}^{K-1} w_{y,k} [\![ y^{(k)}f(\mathbf{x}^{(k)}) \leq 0 ]\!] = c_y \cdot \mathop{\mathbb{E}}_{k\sim P_k} [\![ y^{(k)}f(\mathbf{x}^{(k)}) \leq 0 ]\!],$$

where $P_k(k \mid y) = w_{y,k}/c_y$ is a probability distribution because $c_y = \sum_{k=1}^{K-1} w_{y,k}$. Equivalently, we can define a distribution $\hat{P}(\mathbf{x}^{(k)}, y^{(k)})$ that generates $(\mathbf{x}^{(k)}, y^{(k)})$ by drawing the tuple $(\mathbf{x}, y, k)$ from $P(\mathbf{x}, y)$ and $P_k(k \mid y)$. Then, the generalization error of $r$ is

$$\mathop{\mathbb{E}}_{(\mathbf{x},y)\sim P} \mathcal{C}_{y,r(\mathbf{x})} \leq \mathop{\mathbb{E}}_{(\mathbf{x},y)\sim P} c_y \cdot \mathop{\mathbb{E}}_{k\sim P_k} [\![ y^{(k)}f(\mathbf{x}^{(k)}) \leq 0 ]\!] \leq c \cdot \mathop{\mathbb{E}}_{(\mathbf{x}^{(k)},y^{(k)})\sim \hat{P}} [\![ y^{(k)}f(\mathbf{x}^{(k)}) \leq 0 ]\!]. \quad (8)$$

$\blacksquare$

Theorem 3 shows that, if the binary classifier $f$ generalizes well when examples are sampled from $\hat{P}$, the constructed ranking rule would also generalize well. The terms $y^{(k)}f(\mathbf{x}^{(k)})$, which are exactly the margins of the associated binary classifier $f_{\mathrm{b}}(\mathbf{x}, k)$, would be analogously called the *margins* for ordinal regression, and are expected to be positive and large for correct and confident predictions.

Herbrich et al. [5] derived a large-margin bound for an SVM-based thresholded model using pairwise comparisons between examples. However, the bound is complicated because $O(N^2)$ pairs are taken into consideration, and the bound is restricted because it is only applicable to hard-margin cases, i.e., for all $n$, the margins $y_n^{(k)}f(\mathbf{x}_n^{(k)}) \geq \Delta > 0$. Another large-margin bound was derived by Shashua and Levin [2]. However, the bound is not data-dependent, and hence does not fully explain the generalization performance of large-margin ranking rules in reality (for more discussions on data-dependent bounds, see the work of, for example, Bartlett and Shawe-Taylor [10]).

Next, we show how a novel data-dependent bound for SVM-based ordinal regression approaches can be derived from our reduction framework. Our bound includes only $O(KN)$ extended examples, and applies to both hard-margin and soft-margin cases, i.e., the margins $y^{(k)}f(\mathbf{x}^{(k)})$ can be negative. Similar techniques can be used to derive generalization bounds when AdaBoost is the underlying classifier (see the work of Lin and Li [7] for one of such bounds).

**Theorem 4 (data-dependent bound for support vector ordinal regression)** *Assume that*

$$f(\mathbf{x}, k) \in \left\{ f \colon (\mathbf{x}, k) \mapsto \langle \mathbf{u}, \phi(\mathbf{x}) \rangle - \theta_k, \|\mathbf{u}\|^2 + \|\boldsymbol{\theta}\|^2 \leq 1, \|\phi(\mathbf{x})\|^2 + 1 \leq R^2 \right\}.$$

*If $\boldsymbol{\theta}$ is ordered or every row of $\mathcal{C}$ is convex, for any margin criterion $\Delta$, with probability at least $1 - \delta$, every rank rule $r$ based on $f$ has generalization error no more than*

$$\frac{\beta}{N} \cdot \sum_{n=1}^{N} \sum_{k=1}^{K-1} w_n^{(k)} [\![ y_n^{(k)}f(\mathbf{x}_n^{(k)}) \leq \Delta ]\!] + O\left( \frac{\log N}{\sqrt{N}}, \frac{R}{\Delta}, \sqrt{\log \frac{1}{\delta}} \right), \text{ where } \beta = \frac{\max_y c_y}{\min_y c_y}.$$

PROOF Consider the extended training set $\hat{S} = \left\{ (\mathbf{x}_n^{(k)}, y_n^{(k)}) \right\}$, which contains $N(K-1)$ elements. Each element is a possible outcome from the distribution $\hat{P}$ constructed in Theorem 3. Note, however, that these elements are not all independent. Thus, we cannot directly use the whole extended set as i.i.d. outcomes from $\hat{P}$. Nevertheless, some subsets of $\hat{S}$ do contain i.i.d. outcomes from $\hat{P}$. One way to extract such a subset is to choose independent $k_n$ from $P_k(k \mid y_n)$ for each $(\mathbf{x}_n, y_n)$. The subset would be named $T = \left\{ (\mathbf{x}_n^{(k_n)}, y_n^{(k_n)}) \right\}_{n=1}^{N}$.

Bartlett and Shawe-Taylor [10] showed that with probability at least $(1 - \delta/2)$ over the choice of $N$ i.i.d. outcomes from $\hat{P}$, which is the case of $T$,

$$\mathop{\mathbb{E}}_{(\mathbf{x}^{(k)},y^{(k)})\sim \hat{P}} [\![ y^{(k)}f(\mathbf{x}^{(k)}) \leq 0 ]\!] \leq \frac{1}{N} \sum_{n=1}^{N} [\![ y_n^{(k_n)}f(\mathbf{x}_n^{(k_n)}) \leq \Delta ]\!] + O\left( \frac{\log N}{\sqrt{N}}, \frac{R}{\Delta}, \sqrt{\log \frac{1}{\delta}} \right). \quad (9)$$

Table 1: Test error with absolute cost

| data set | Reduction based on | | | SVOR-IMC with kernel | |
|---|---|---|---|---|---|
| | C4.5 | boost-stump | SVM-perceptr. | perceptron | Gaussian [4] |
| pyrimidines | $1.565 \pm 0.072$ | $1.360 \pm 0.054$ | $\mathbf{1.304 \pm 0.040}$ | $\mathbf{1.315 \pm 0.039}$ | $\mathbf{1.294 \pm 0.046}$ |
| machine | $0.987 \pm 0.024$ | $0.875 \pm 0.017$ | $0.842 \pm 0.022$ | $\mathbf{0.814 \pm 0.019}$ | $0.990 \pm 0.026$ |
| boston | $0.950 \pm 0.016$ | $0.846 \pm 0.015$ | $\mathbf{0.732 \pm 0.013}$ | $\mathbf{0.729 \pm 0.013}$ | $0.747 \pm 0.011$ |
| abalone | $1.560 \pm 0.006$ | $1.458 \pm 0.005$ | $1.383 \pm 0.004$ | $1.386 \pm 0.005$ | $\mathbf{1.361 \pm 0.003}$ |
| bank | $1.700 \pm 0.005$ | $1.481 \pm 0.002$ | $1.404 \pm 0.002$ | $1.404 \pm 0.002$ | $\mathbf{1.393 \pm 0.002}$ |
| computer | $0.701 \pm 0.003$ | $0.604 \pm 0.002$ | $\mathbf{0.565 \pm 0.002}$ | $\mathbf{0.565 \pm 0.002}$ | $0.596 \pm 0.002$ |
| california | $0.974 \pm 0.004$ | $0.991 \pm 0.003$ | $\mathbf{0.940 \pm 0.001}$ | $\mathbf{0.939 \pm 0.001}$ | $1.008 \pm 0.001$ |
| census | $1.263 \pm 0.003$ | $1.210 \pm 0.001$ | $\mathbf{1.143 \pm 0.002}$ | $\mathbf{1.143 \pm 0.002}$ | $1.205 \pm 0.002$ |

Let $b_n = [\![ y_n^{(k_n)} f(\mathbf{x}_n^{(k_n)}) \leq \Delta ]\!]$ be a Boolean random variable introduced by $k_n \sim P_k(k \mid y_n)$. The variable has mean $c_{y_n}^{-1} \cdot \sum_{k=1}^{K-1} w_n^{(k)} [\![ y_n^{(k)} f(\mathbf{x}_n^{(k)}) \leq \Delta ]\!]$. An extended Chernoff bound shows that when each $b_n$ is chosen independently, with probability at least $(1 - \delta/2)$ over the choice of $b_n$,

$$\frac{1}{N} \sum_{n=1}^{N} b_n \leq \frac{1}{N} \sum_{n=1}^{N} \frac{1}{c_{y_n}} \sum_{k=1}^{K-1} w_n^{(k)} [\![ y_n^{(k)} f(\mathbf{x}_n^{(k)}) \leq \Delta ]\!] + O \left( \frac{1}{\sqrt{N}}, \sqrt{\log \frac{1}{\delta}} \right). \qquad (10)$$

The desired result can be obtained by combining (8), (9), and (10) with a union bound. ∎

## 5 Experiments

We performed experiments with eight benchmark data sets that were used by Chu and Keerthi [4]. The data sets were produced by quantizing some metric regression data sets with $K = 10$. We used the same training/test ratio and also averaged the results over 20 trials. Thus, with the absolute cost matrix, we can fairly compare our results with those of SVOR-IMC [4].

We tested our framework with $\mathbf{E} = \gamma \mathbf{I}_{K-1}$ and three different binary classification algorithms. The first binary algorithm is Quinlan's C4.5 [11]. The second is AdaBoost-stump which uses Ada-Boost to aggregate 500 decision stumps. The third one is SVM with the perceptron kernel [12], with a simple setting of $\gamma = 1$. Note that the Gaussian kernel was used by Chu and Keerthi [4]. We used the perceptron kernel instead to gain the advantage of faster parameter selection. The parameter $\kappa$ of the soft-margin SVM was determined by a 5-fold cross validation procedure with $\log_2 \kappa = -17, -15, \ldots, 3$, and LIBSVM [13] was adopted as the solver. For a fair comparison, we also implemented SVOR-IMC with the perceptron kernel and the same parameter selection procedure in LIBSVM.

We list the mean and the standard error of all test results in Table 1, with entries within one standard error of the lowest one marked in bold. With our reduction framework, all the three binary learning algorithms could be better than SVOR-IMC with the Gaussian kernel on some of the data sets, which demonstrates that they achieve decent out-of-sample performances. Among the three algorithms, SVM-perceptron is significantly better than the other two.

Within the three SVM-based approaches, the two with the perceptron kernel are better than SVOR-IMC with the Gaussian kernel in test performance. Our direct reduction to the standard SVM performs similarly to SVOR-IMC with the same perceptron kernel, but is much easier to implement. In addition, our direct reduction is significantly faster than SVOR-IMC in training, which is illustrated in Figure 1 using the four largest data sets.[4] The main cause to the time difference is the speedup heuristics. While, to the best of our knowledge, not much

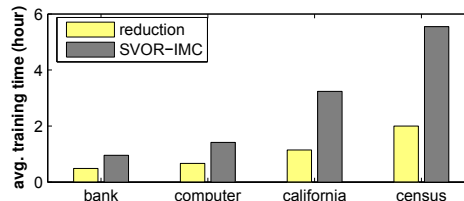

Figure 1: Training time (including automatic parameter selection) of the SVM-based approaches with the perceptron kernel

has been done to improve the original SVOR-IMC algorithm, plenty of heuristics, such as shrinking and advanced working set selection in LIBSVM, can be seamlessly adopted by our direct reduction. This difference demonstrates another advantage of our reduction framework: improvements to binary classification approaches can be immediately inherited by reduction-based ordinal regression algorithms.

# 6   Conclusion

We presented a reduction framework from ordinal regression to binary classification based on extended examples. The framework has the flexibility to work with any reasonable cost matrix and any binary classifiers. We demonstrated the algorithmic advantages of the framework in designing new ordinal regression algorithms and explaining existing algorithms. We also showed that the framework can be used to derive new generalization bounds for ordinal regression. Furthermore, the usefulness of the framework was empirically validated by comparing three new algorithms constructed from our framework with the state-of-the-art SVOR-IMC algorithm.

**Acknowledgments**

We wish to thank Yaser S. Abu-Mostafa, Amrit Pratap, John Langford, and the anonymous reviewers for valuable discussions and comments. Ling Li was supported by the Caltech SISL Graduate Fellowship, and Hsuan-Tien Lin was supported by the Caltech EAS Division Fellowship.

**References**

[1] K. Crammer and Y. Singer. Pranking with ranking. In T. G. Dietterich, S. Becker, and Z. Ghahramani, eds., *Advances in Neural Information Processing Systems 14*, vol. 1, pp. 641–647. MIT Press, 2002.

[2] A. Shashua and A. Levin. Ranking with large margin principle: Two approaches. In S. Becker, S. Thrun, and K. Obermayer, eds., *Advances in Neural Information Processing Systems 15*, pp. 961–968. MIT Press, 2003.

[3] S. Rajaram, A. Garg, X. S. Zhou, and T. S. Huang. Classification approach towards ranking and sorting problems. In N. Lavrač, D. Gamberger, H. Blockeel, and L. Todorovski, eds., *Machine Learning: ECML 2003*, vol. 2837 of *Lecture Notes in Artificial Intelligence*, pp. 301–312. Springer-Verlag, 2003.

[4] W. Chu and S. S. Keerthi. New approaches to support vector ordinal regression. In L. D. Raedt and S. Wrobel, eds., *ICML 2005: Proceedings of the 22nd International Conference on Machine Learning*, pp. 145–152. Omnipress, 2005.

[5] R. Herbrich, T. Graepel, and K. Obermayer. Large margin rank boundaries for ordinal regression. In A. J. Smola, P. L. Bartlett, B. Schölkopf, and D. Schuurmans, eds., *Advances in Large Margin Classifiers*, chapter 7, pp. 115–132. MIT Press, 2000.

[6] E. Frank and M. Hall. A simple approach to ordinal classification. In L. D. Raedt and P. Flach, eds., *Machine Learning: ECML 2001*, vol. 2167 of *Lecture Notes in Artificial Intelligence*, pp. 145–156. Springer-Verlag, 2001.

[7] H.-T. Lin and L. Li. Large-margin thresholded ensembles for ordinal regression: Theory and practice. In J. L. Balcázar, P. M. Long, and F. Stephan, eds., *Algorithmic Learning Theory: ALT 2006*, vol. 4264 of *Lecture Notes in Artificial Intelligence*, pp. 319–333. Springer-Verlag, 2006.

[8] Y. Freund and R. E. Schapire. Large margin classification using the perceptron algorithm. *Machine Learning*, 37(3):277–296, 1999.

[9] V. N. Vapnik. *The Nature of Statistical Learning Theory*. Springer-Verlag, 2nd edition, 1999.

[10] P. Bartlett and J. Shawe-Taylor. Generalization performance of support vector machines and other pattern classifiers. In B. Schölkopf, C. J. C. Burges, and A. J. Smola, eds., *Advances in Kernel Methods: Support Vector Learning*, chapter 4, pp. 43–54. MIT Press, 1998.

[11] J. R. Quinlan. Induction of decision trees. *Machine Learning*, 1(1):81–106, 1986.

[12] H.-T. Lin and L. Li. Novel distance-based SVM kernels for infinite ensemble learning. In *Proceedings of the 12th International Conference on Neural Information Processing*, pp. 761–766, 2005.

[13] C.-C. Chang and C.-J. Lin. *LIBSVM: A library for support vector machines*, 2001. Software available at `http://www.csie.ntu.edu.tw/~cjlin/libsvm`.

## Footnotes

[1]The Boolean test $[\![\cdot]\!]$ is 1 if the inner condition is true, and 0 otherwise.

[2]To precisely replicate the PRank algorithm, the $(K-1)$ extended examples sprouted from a same example should be considered altogether in updating the perceptron weight vector.

[3]The formulation was only briefly mentioned in a footnote, but not studied, by Chu and Keerthi [4].

[4]The results are averaged CPU time gathered on a 1.7G Dual Intel Xeon machine with 1GB of memory.
